# Principled Architecture Selection for Neural Networks: Application to Corporate Bond Rating Prediction

**John Moody**
Department of Computer Science
Yale University
P. O. Box 2158 Yale Station
New Haven, CT 06520

**Joachim Utans**
Department of Electrical Engineering
Yale University
P. O. Box 2157 Yale Station
New Haven, CT 06520

## Abstract

The notion of generalization ability can be defined precisely as the *prediction risk*, the expected performance of an estimator in predicting new observations. In this paper, we propose the prediction risk as a measure of the generalization ability of multi–layer perceptron networks and use it to select an optimal network architecture from a set of possible architectures. We also propose a heuristic search strategy to explore the space of possible architectures. The prediction risk is estimated from the available data; here we estimate the prediction risk by *v–fold cross–validation* and by asymptotic approximations of *generalized cross–validation* or Akaike's *final prediction error*. We apply the technique to the problem of predicting corporate bond ratings. This problem is very attractive as a case study, since it is characterized by the limited availability of the data and by the lack of a complete *a priori* model which could be used to impose a structure to the network architecture.

## 1 Generalization and Prediction Risk

The notion of generalization ability can be defined precisely as the *prediction risk*, the expected performance of an estimator is predicting new observations. Consider a set of observations $D = \{(\vec{x}_j, t_j); j = 1 \ldots N\}$ that are assumed to be generated

as

$$t_j = \mu(x_j) + \epsilon_j \tag{1}$$

where $\mu(x)$ is an unknown function, the inputs $x_j$ are drawn independently with an unknown stationary probability density function $p(x)$, the $\epsilon_j$ are independent random variables with zero mean ($\bar{\epsilon} = 0$) and variance $\sigma_\epsilon^2$, and the $t_j$ are the observed target values.

The learning or regression problem is to find an estimate $\hat{\mu}_\lambda(x; D)$ of $\mu(x)$ given the data set $D$ from a class of predictors or models $\mu_\lambda(x)$ indexed by $\lambda$. In general, $\lambda \in \Lambda = (S, A, W)$, where $S \subset X$ denotes a chosen subset of the set of available input variables $X$, $A$ is a selected architecture within a class of model architectures $\mathcal{A}$, and $W$ are the adjustable parameters (weights) of architecture $A$.

The *prediction risk* $P(\lambda)$ is defined as the expected performance on future data and can be approximated by the expected performance on a finite test set:

$$P(\lambda) = \int dx \, p(x)[\mu(x) - \hat{\mu}(x)]^2 + \sigma_\epsilon^2 \approx E\{\frac{1}{N}\sum_{j=1}^{N}(t_j^* - \hat{\mu}_\lambda(x_j^*))^2\} \tag{2}$$

where $(x_j^*, t_j^*)$ are new observations that were not used in constructing $\hat{\mu}_\lambda(x)$. In what follows, we shall use $P(\lambda)$ as a measure of the generalization ability of a model. See [4] and [6] for more detailed presentations.

## 2    Estimates of Prediction Risk

Since we cannot directly calculate the prediction risk $P_\lambda$, we have to estimate it from the available data $D$. The standard method based on test–set validation is not advisable when the data set is small. In this paper we consider such a case; the prediction of corporate bond ratings from a database of only 196 firms. Cross-validation (CV) is a sample re–use method for estimating prediction risk; it makes maximally efficient use of the available data. Other methods are the generalized cross–validation (GCV) and the final prediction error (FPE) criteria, which combine the average training squared error $ASE$ with a measure of the model complexity. These will be discussed in the next sections.

### 2.1    Cross Validation

Cross–Validation is a method that makes minimal assumptions on the statistics of the data. The idea of cross validation can be traced back to Mosteller and Tukey [7]. For reviews, see Stone [8, 9], Geisser [5] and Eubank [4].

Let $\hat{\mu}_{\lambda(j)}(x)$ be a predictor trained using all observations except $(x_j, t_j)$ such that $\hat{\mu}_{\lambda(j)}(x)$ minimizes

$$ASE_j = \frac{1}{(N-1)} \sum_{k \neq j} \left(t_k - \hat{\mu}_{\lambda(j)}(x_k)\right)^2$$

Then, an estimator for the prediction risk $P(\lambda)$ is the *cross validation average*

*squared error*

$$CV(\lambda) = \frac{1}{N} \sum_{j=1}^{N} \left(t_j - \hat{\mu}_{\lambda(j)}(x_j)\right)^2 \qquad (3)$$

This form of $CV(\lambda)$ is known as *leave–one–out* cross–validation.

However, $CV(\lambda)$ in (3) is expensive to compute for neural network models; it involves constructing $N$ networks, each trained with $N-1$ patterns. For the work described in this paper we therefore use a variation of the method, *v-fold cross–validation*, that was introduced by Geisser [5] and Wahba *et al* [12]. Instead of leaving out only one observation for the computation of the sum in (3) we delete larger subsets of $D$.

Let the data $D$ be divided into $v$ randomly selected disjoint subsets $P_j$ of roughly equal size: $\cup_{j=1}^{v} P_j = D$ and $\forall i \neq j$, $P_i \cap P_j = \emptyset$. Let $N_j$ denote the number of observations in subset $P_j$. Let $\hat{\mu}_{\lambda(P_j)}(x)$ be an estimator trained on all data except for $(x,t) \in P_j$. Then, the cross-validation average squared error for subset $j$ is defined as

$$CV_{P_j}(\lambda) = \frac{1}{N_j} \sum_{(x_k,t_k)\in P_j} \left(t_k - \hat{\mu}_{\lambda(P_j)}(x_k)\right)^2,$$

and

$$CV_P(\lambda) = \frac{1}{v} \sum_{j} CV_{P_j}(\lambda). \qquad (4)$$

Typical choices for $v$ are 5 and 10. Note that leave–one–out $CV$ is obtained in the limit $v = N$.

## 2.2  Generalized Cross-Validation and Final Prediction Error

For linear models, two useful criteria for selecting a model architecture are *generalized cross–validation (GCV)* (Wahba [11]) and Akaike's *final prediction error (FPE)* ([1]):

$$GCV(\lambda) = ASE(\lambda)\frac{1}{\left(1 - \frac{S(\lambda)}{N}\right)^2} \qquad FPE(\lambda) = ASE(\lambda)\left(\frac{1 + \frac{S(\lambda)}{N}}{1 - \frac{S(\lambda)}{N}}\right).$$

$S(\lambda)$ denotes the number of weights of model $\lambda$. See [4] for a tutorial treatment. Note that although they are slightly different for small sample sizes, they are asymptotically equivalent for large $N$:

$$\hat{P}(\lambda) \equiv ASE(\lambda)\left(1 + 2\frac{S(\lambda)}{N}\right) \approx GCV(\lambda) \approx FPE(\lambda) \qquad (5)$$

We shall use this asymptotic estimate for the prediction risk in our analysis of the bond rating models.

It has been shown by Moody [6] that FPE and therefore $\hat{P}(\lambda)$ is an unbiased estimate of the prediction risk for the neural network models considered here provided that (1) the noise $\epsilon_j$ in the observed targets $t_j$ is independent and identically distributed,

(2) weight decay is not used, and (3) the resulting model is unbiased. (In practice, however, essentially all neural network fits to data will be biased (see Moody [6]).) FPE is a special case of Barron's PSE [2] and Moody's GPE [6]. Although FPE and $\hat{P}(\lambda)$ are unbiased only under the above assumptions, they are much cheaper to compute than $CV_P$ since no retraining is required.

# 3   A Case Study: Prediction of Corporate Bond Ratings

A bond is a debt security which constitutes a promise by the issuing firm to pay a given rate of interest on the original issue price and to redeem the bond at face value at maturity. Bonds are rated according to the default risk of the issuing firm by independent rating agencies such as Standard & Poors (S&P) and Moody's Investor Service. The firm is in default if it is not able make the promised interest payments.

| Representation of S&P Bond Ratings | | | | | | | | | | | | |
|---|---|---|---|---|---|---|---|---|---|---|---|---|
| CCC | B- | B | ... | BBB+ | A- | A | A+ | AA- | AA | AA+ | AAA- | AAA |
| 2 | 3 | 4 | ... | 11 | 12 | 13 | 14 | 15 | 16 | 17 | 18 | 19 |
| high default risk | | | | ......... | | | | | | low default risk | | |

Table 1: Key to S&P bond ratings. We only used the range from 'AAA' or 'very low default risk' to 'CCC' meaning 'very high default risk'. (Note that AAA- is a not a standard category; its inclusion was suggested to us by a Wall Street analyst.) Bonds with rating BBB- or better are "investment grade" while "junk bonds" have ratings BB+ or below. For our output representation, we assigned an integer number to each rating as shown.

S&P and Moody's determine the rating from various financial variables and possibly other information, but the exact set of variables is unknown. It is commonly believed that the rating is at least to some degree judged on the basis of subjective factors and on variables not directly related to a particular firm. In addition, the method used for assigning the rating based on the input variables is unknown. The problem we are considering here is to predict the S&P rating of a bond based on fundamental financial information about the issuer which is publicly available. Since the rating agencies update their bond ratings infrequently, there is considerable value to being able to anticipate rating changes before they are announced. A predictive model which maps fundamental financial factors onto an estimated rating can accomplish this.

The input data for our model consists of 10 financial ratios reflecting the fundamental characteristics of the firms. The database was prepared for us by analysts at a major financial institution. Since we did not attempt to include all information in the input variables that could possibly be related to a firms bond rating (e.g. all fundamental or technical financial factors, or qualitative information such as quality of management), we can only attempt to approximate the S&P rating.

## 3.1   A Linear Bond Rating Predictor

For comparison with the neural network models, we computed a standard linear regression model. All input variables were used to predict the rating which is represented by a number in $[0, 1]$. The rating varies continuously from one category to the next higher or next lower one and this "smoothness" is captured in the single output representation and should make the task easier. To interpret the network

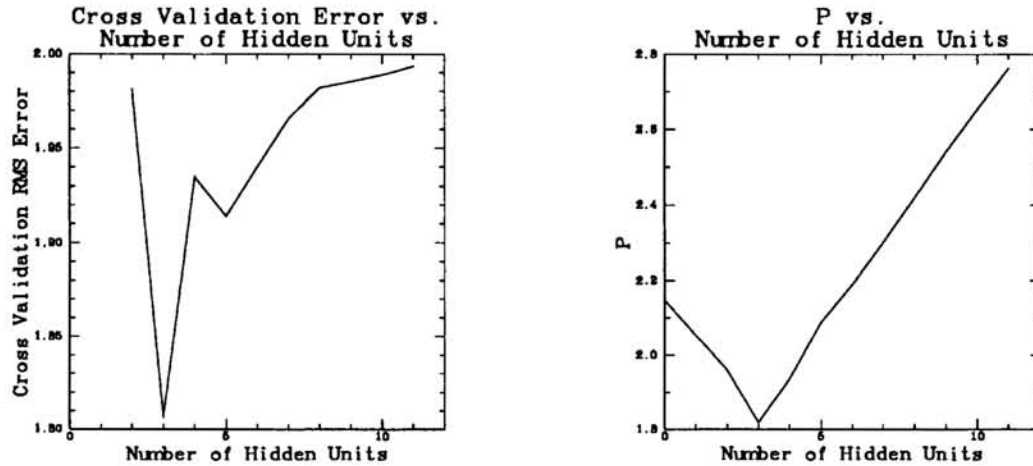

Figure 1: Cross validation error $CV_P(\lambda)$ and $\hat{P}(\lambda)$ versus number of hidden units.

response, the output was rescaled from $[0,1]$ to $[2,19]$ and rounded to the nearest integer; 19 corresponds to a rating of 'AAA' and 2 to 'CCC' and below (see Table 1). The input variables were normalized to the interval $[0,1]$ since the original financial ratios differed widely in magnitude. The model predicted the rating of 21.4 % of the firms correctly, for 37.2 % the error was one notch and for 21.9 % two notches (thus predicting 80.5 % of the data within two notches from the correct target). The RMS training error was 1.93 and the estimate of the prediction risk $\hat{P} = 2.038$.

## 3.2 Beyond Linear Regression: Prediction by Two Layer Perceptrons

The class of models we are considering as predictors are two-layer perceptron networks with $I_\lambda$ input variables, $H_\lambda$ internal units and a single output unit having the form

$$\hat{\mu}_\lambda(x) = f\left(v_0 + \sum_{\alpha=1}^{H_\lambda} v_\alpha\, g\left(w_{\alpha 0} + \sum_{\beta=1}^{I_\lambda} w_{\alpha\beta}\, x_\beta\right)\right). \qquad (6)$$

The hidden units have a sigmoidal transfer function while our single output unit uses a piecewise linear function.

## 3.3 Heuristic Search over the Space of Perceptron Architectures

Our proposed heuristic search algorithm over the space of perceptron architectures is as follows. First, we select the optimal number of internal units from a sequence of fully connected networks with increasing number of hidden units. Then, using the optimal fully connected network, we prune weights and input variables in parallel resulting in two separately pruned networks. Lastly, the methods were combined and the resulting networks is retrained to yield the final model

### 3.3.1 Selecting the Number of Hidden Units

We initially trained fully connected networks with all 10 available inputs variables but with the number of hidden units $H_\lambda$ varying from 2 to 11. Five–fold cross–

| Training Error 3 Hidden Units | | | |
|---|---|---|---|
| $|E_{notch}|$ | firms | % | cum. % |
| 0 | 67 | 34.2 | 34.2 |
| 1 | 84 | 42.9 | 77.1 |
| 2 | 34 | 17.3 | 94.4 |
| > 2 | 11 | 5.6 | 100.0 |
| number of weights | | | 37 |
| standard deviation | | | 1.206 |
| mean absolute deviation | | | 0.898 |
| training error | | | 1.320 |

| Cross Validation Error 3 Hidden Units | | | |
|---|---|---|---|
| $|E_{notch}|$ | firms | % | cum. % |
| 0 | 54 | 28.6 | 28.6 |
| 1 | 77 | 38.8 | 67.3 |
| 2 | 35 | 17.3 | 84.7 |
| > 2 | 30 | 15.3 | 100.0 |
| number of weights | | | 37 |
| standard deviation | | | 1.630 |
| mean absolute deviation | | | 1.148 |
| cross validation error | | | 1.807 |

Table 2: Results for the network with 3 hidden units. The standard deviation and the mean absolute deviation are computed after rescaling the output of the network to [2,19] and rounding to the nearest integer (notches). The RMS training error is computed using the rescaled output of the network before rounding. The table also describes the predictive ability of the network by a histogram; the error column gives the number of rating categories the network was off from the correct target. The network with 3 hidden units significantly outperformed the linear regression model. On the right Cross Validation results for the network with 3 hidden units are shown. In order to predict the rating for a firm, we choose among the networks trained for the cross-validation procedure the one that was not trained using the subset the firm belongs to. Thus the results concerning the predictive ability of the model reflect the expected performance of the model trained on all the data with new data in the cross–validation–sense.

validation and $\hat{P}(\lambda)$ were used to select the number of hidden units. We compute $CV_P(\lambda)$ according to equation (4); the data set was partitioned into $v = 5$ subsets. We also computed $\hat{P}(\lambda)$ according to equation (5). The results of the two methods are consistent, having a common minimum for $H_\lambda = 3$ internal units (see figure 1).

Table 2(left) shows the results for the network with $H_\lambda = 3$ trained on the entire data set. A more accurate description of the performance of the model is shown in table 2(right) were the predictive ability is calculated from the hold–out sets of the cross–validation procedure.

### 3.3.2   Pruning of Input Variables via Sensitivity Analysis

Next, we attempted to further reduce the number of weights of the network by eliminating some of the input variables. To test which inputs are most significant for determining the network output, we perform a sensitivity analysis. We define the "Sensitivity" of the network model to variable $\beta$ as:

$$S_\beta = \frac{1}{N} \sum_{j=1}^{N} ASE(\overline{x}_\beta) - ASE(x_\beta) \quad \text{with} \quad \overline{x}_\beta = \frac{1}{N} \sum_{j=1}^{N} x_{\beta_j} .$$

Here, $x_{\beta_j}$ is the $\beta^{th}$ input variable of the $j^{th}$ exemplar. $S_\beta$ measures the effect on the training $ASE$ of replacing the $\beta^{th}$ input $x_\beta$ by its average $\overline{x}_\beta$. Replacement of a variable by its average value removes its influence on the network output. Again we use 5-fold cross–validation and $\hat{P}$ to estimate the prediction risk $P_\lambda$. We constructed a sequence of models by deleting an increasing number of input variables in order of increasing $S_\beta$. For each model, $CV_P$ and $\hat{P}$ was computed, figure 2 shows the results. A minimum was attained for the model with $I_\lambda = 8$ input variables (2 inputs were removed). This reduces the number of weights by $2H_\lambda = 6$.

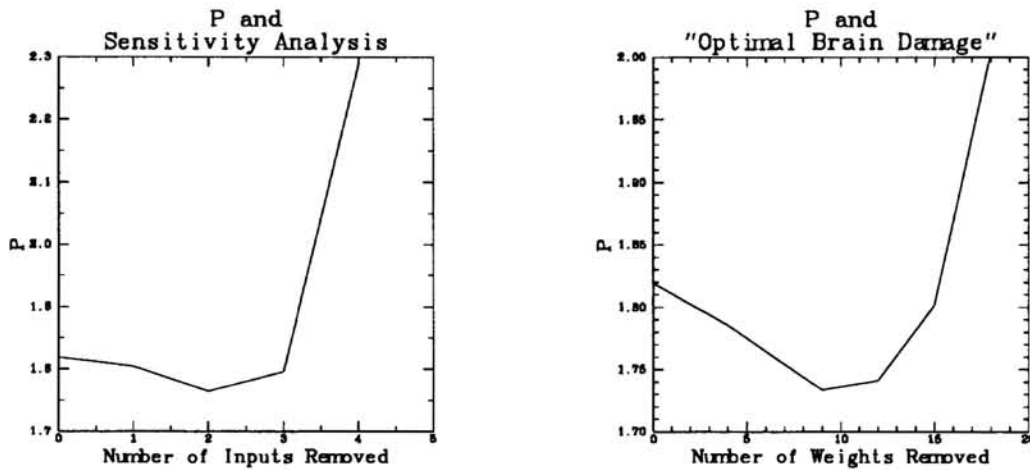

Figure 2: $\hat{P}(\lambda)$ for the sensitivity analysis and OBD. In both cases, the Cross validation error $CV_P(\lambda)$ has a minimum for the same $\lambda$.

### 3.3.3   Weight Pruning via "Optimal Brain Damage"

*Optimal Brain Damage (OBD)* was introduced by Le Cun *at al* [3] as a method to reduce the number of weights in a neural network to avoid overfitting. OBD is designed to select those weights in the network whose removal will have a small effect on the training *ASE*. Assuming that the original network was too large, removing these weights and retraining the now smaller network should improve the generalization performance. The method approximates *ASE* at a minimum in weight space by a diagonal quadratic expansion. The saliency

$$s_i = \frac{1}{2} \frac{\partial^2 ASE}{\partial w_i^2} w_i^2$$

computed after training has stopped is a measure (in the diagonal approximation) for the change of *ASE* when weight $w_i$ is removed from the network.

$CV_P$ and $\hat{P}$ were computed to select the optimal model. We find that $CV_P$ and $\hat{P}$ are minimized when 9 weights are deleted from the network using all input variables. However, some overlap exists when compared to the sensitivity analysis described above: 5 of the deleted weights would also have been removed by the sensitivity method.

Table 3 show the overall performance of our model when the two techniques were combined to yield the final architecture. This architecture is obtained by deleting the union of the sets of weights that were deleted using weight and input pruning separately. Note the improvement in estimated prediction performance (CV error) in table 3 relative to 2.

## 4   Summary

Our example shows that (1) nonlinear network models can out–perform linear regression models, and (2) substantial benefits in performance can be obtained by the use of principled architecture selection methods. The resulting structured networks

| Training Error, 3 Hidden Units 2 Inputs and 9 Connections Removed | | | |
|---|---|---|---|
| $|E_{notch}|$ | firms | % | cum. % |
| 0 | 69 | 35.2 | 35.2 |
| 1 | 81 | 41.3 | 76.5 |
| 2 | 32 | 16.3 | 92.8 |
| >2 | 14 | 7.2 | 100.0 |
| number of weights | | 27 | |
| standard deviation | | 1.208 | |
| mean absolute deviation | | 0.882 | |
| training error | | 1.356 | |

| Cross Validation Error, 3 Hidden Units 2 Inputs and 9 Connections Removed | | | |
|---|---|---|---|
| $|E_{notch}|$ | firms | % | cum. % |
| 0 | 58 | 29.6 | 29.6 |
| 1 | 76 | 38.8 | 68.4 |
| 2 | 37 | 18.9 | 87.2 |
| >2 | 26 | 12.8 | 100.0 |
| number of weights | | 27 | |
| standard deviation | | 1.546 | |
| mean absolute deviation | | 1.117 | |
| cross validation error | | 1.697 | |

Table 3: Results for the network with 3 hidden units with both, sensitivity analysis and OBD applied. Note the improvement in CV error performance of relative to Table 2.

are optimized with respect to the task at hand, even though it may not be possible to design them based on *a priori* knowledge.

Estimates of the prediction risk offer a sound basis for assessing the performance of the model on new data and can be used as a tool for principled architecture selection. Cross–validation, GCV and FPE provide computationally feasible means of estimating the prediction risk. These estimates of prediction risk provide very effective criteria for selecting the number of internal units and performing sensitivity analysis and OBD.

**References**

[1] H. Akaike. Statistical predictor identification. *Ann. Inst. Statist. Math.*, 22:203–217, 1970.

[2] A. Barron. Predicted squared error: a criterion for automatic model selection. In S. Farlow, editor, *Self–Organizing Methods in Modeling*. Marcel Dekker, New York, 1984.

[3] Y. Le Cun, J. S. Denker, and S. A. Solla. Optimal brain damage. In D. S. Touretzky, editor, *Advances in Neural Information Processing Systems 2*. Morgan Kaufmann Publishers, 1990.

[4] Randall L. Eubank. *Spline Smoothing and Nonparametric Regression*. Marcel Dekker, Inc., 1988.

[5] Seymour Geisser. The predictive sample reuse method with applications. *Journal of The American Statistical Association*, 70(350), June 1975.

[6] John Moody. The effective number of parameters: an analysis of generalization and regularization in nonlinear learning systems. *short version in this volume, long version to appear*, 1992.

[7] F. Mosteller and J. W. Tukey. Data analysis, including statistics. In G. Lindzey and E. Aronson, editors, *Handbook of Social Psychology, Vol. 2*. Addison–Wesley, 1968 (first edition 1954).

[8] M. Stone. Cross–validatory choice and assessment of statistical predictions. *Roy. Stat. Soc.*, B36, 1974.

[9] M. Stone. Cross–validation: A review. *Math. Operationsforsch. Statist., Ser. Statistics*, 9(1), 1978.

[10] Joachim Utans and John Moody. Selecting neural network architectures via the prediction risk: Application to corporate bond rating prediction. In *Proceedings of the First International Conference on Artifical Intelligence Applications on Wall Street*. IEEE Computer Society Press, Los Alamitos, CA, 1991.

[11] G. Wahba. *Spline Models for Observational Data*, volume 59 of *Regional Conference Series in Applied Mathematics*. SIAM Press, Philadelphia, 1990.

[12] G. Wahba and S. Wold. A completely automatic french curve: Fitting spline functions by cross–validation. *Communiations in Statistics*, 4(1):1–17, 1975.
